# On Local Rewards and Scaling Distributed Reinforcement Learning

**J. Andrew Bagnell**
Robotics Institute
Carnegie Mellon University
Pittsburgh, PA 15213
dbagnell@ri.cmu.edu

**Andrew Y. Ng**
Computer Science Department
Stanford University
Stanford, CA 94305
ang@cs.stanford.edu

## Abstract

We consider the scaling of the number of examples necessary to achieve good performance in distributed, cooperative, multi-agent reinforcement learning, as a function of the the number of agents $n$. We prove a worst-case lower bound showing that algorithms that rely solely on a *global* reward signal to learn policies confront a fundamental limit: They require a number of real-world examples that scales roughly linearly in the number of agents. For settings of interest with a very large number of agents, this is impractical. We demonstrate, however, that there is a class of algorithms that, by taking advantage of *local* reward signals in large distributed Markov Decision Processes, are able to ensure good performance with a number of samples that scales as $O(\log n)$. This makes them applicable even in settings with a very large number of agents $n$.

## 1 Introduction

Recently there has been great interest in distributed reinforcement learning problems where a collection of agents with independent action choices attempts to optimize a joint performance metric. Imagine, for instance, a traffic engineering application where each traffic signal may independently decide when to switch colors, and performance is measured by aggregating the throughput at all traffic stops. Problems with such factorizations where the *global* reward decomposes in to a sum of *local* rewards are common and have been studied in the RL literature. [10]

The most straightforward and common approach to solving these problems is to apply one of the many well-studied single agent algorithms to the global reward signal. Effectively, this treats the multi-agent problem as a single agent problem with a very large action space. Peshkin et al. [9] establish that policy gradient learning factorizes into independent policy gradient learning problems for each agent using the global reward signal. Chang et al. [3] use global reward signals to estimate effective local rewards for each agent. Guestrin et al. [5] consider coordinating agent actions using the global reward. We argue from an information theoretic perspective that such algorithms are fundamentally limited in their scalability. In particular, we show in Section 3 that as a function of the number of agents $n$, such algorithms will need to see[1] $\tilde{\Omega}(n)$ trajectories in the worst case to achieve good performance.

We suggest an alternate line of inquiry, pursued as well by other researchers (including

notably [10]), of developing algorithms that capitalize on the availability of local reward signals to improve performance. Our results show that such local information can dramatically reduce the number of examples necessary for learning to $O(\log n)$. One approach that the results suggest to solving such distributed problems is to estimate model parameters from all local information available, and then to solve the resulting model offline. Although this clearly still carries a high *computational* burden, it is much preferable to requiring a large amount of real-world experience. Further, useful approximate multiple agent Markov Decision Process (MDP) solvers that take advantage of local reward structure have been developed. [4]

## 2  Preliminaries

We consider distributed reinforcement learning problems, modeled as MDPs, in which there are $n$ (cooperative) agents, each of which can directly influence only a small number of its neighbors. More formally, let there be $n$ agents, each with a finite state space $S$ of size $|S|$ states and a finite action space $A$ of size $|A|$. The joint state space of all the agents is therefore $S^n$, and the joint action space $A^n$. If $s_t \in S^n$ is the joint state of the agents at time $t$, we will use $s_t^{(i)}$ to denote the state of agent $i$. Similarly, let $a_t^{(i)}$ denote the action of agent $i$.

For each agent $i \in \{1, \ldots, n\}$, we let $\mathrm{neigh}(i) \subseteq \{1, \ldots, n\}$ denote the subset of agents that $i$'s state directly influences. For notational convenience, we assume that if $i \in \mathrm{neigh}(j)$, then $j \in \mathrm{neigh}(i)$, and that $i \in \mathrm{neigh}(i)$. Thus, the agents can be viewed as living on the vertices of a graph, where agents have a direct influence on each other's state only if they are connected by an edge. This is similar to the graphical games formalism of [7], and is also similar to the Dynamic Bayes Net (DBN)-MDP formalisms of [6] and [2]. (Figure 1 depicts a DBN and an agent influence graph.) DBN formalisms allow the more refined notion of directionality in the influence between neighbors.

More formally, each agent $i$ is associated with a CPT (conditional probability table) $P_i(s_{t+1}^{(i)}|s_t^{(\mathrm{neigh}(i))}, a_t^{(i)})$, where $s_t^{(\mathrm{neigh}(i))}$ denotes the state of agent $i$'s neighbors at time $t$. Given the joint action $a$ of the agents, the joint state evolves according to

$$p(s_{t+1}|s_t, a_t) = \prod_{i=1}^{n} p(s_{t+1}^{(i)}|s_t^{(\mathrm{neigh}(i))}, a_t^{(i)}). \tag{1}$$

For simplicity, we have assumed that agent $i$'s state is directly influenced by the states of $\mathrm{neigh}(i)$ but not their actions; the generalization offers no difficulties. The initial state $s_1$ is distributed according to some initial-state distribution $\mathcal{D}$.

A policy is a map $\pi : S^n \mapsto A^n$. Writing $\pi$ out explicitly as a vector-valued function, we have $\pi(s) = (\pi_1(s), \ldots, \pi_n(s))$, where $\pi_i(s) : S^n \mapsto A$ is the local policy of agent $i$. For some applications, we may wish to consider only policies in which agent $i$ chooses its local action as a function of only its local state $s^{(i)}$ (and possibly its neighbors); in this case, $\pi_i$ can be restricted to depend only on $s^{(i)}$.

Each agent has a **local reward function** $R_i(s^{(i)}, a^{(i)})$, which takes values in the unit interval $[0, 1]$. The total payoff in the MDP at each step is $R(s, a) = (1/n) \sum_{i=1}^{n} R(s^{(i)}, a^{(i)})$. We call this $R(s, a)$ the **global reward function**, since it reflects the total reward received by the joint set of agents. We will consider the finite-horizon setting, in which the MDP terminates after $T$ steps. Thus, the utility of a policy $\pi$ in an MDP $M$ is

$$U(\pi) = U_M(\pi) = \mathrm{E}_{s_1 \sim \mathcal{D}}[V^\pi(s_1)] = \mathrm{E}\left[\frac{1}{n} \sum_{t=1}^{T} \sum_{i=1}^{n} R_i(s_t^{(i)}, a_t^{(i)})|\pi\right].$$

In the reinforcement learning setting, the dynamics (CPTs) and rewards of the problem are unknown, and a learning algorithm has to take actions in the MDP and use the resulting observations of state transitions and rewards to learn a good policy. Each "trial" taken by a reinforcement learning algorithm shall consist of a $T$-step sequence in the MDP.

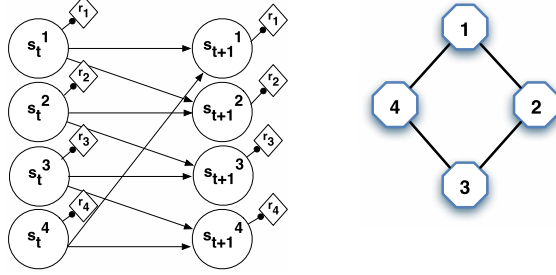

Figure 1: (Left) A DBN description of a multi-agent MDP. Each row of (round) nodes in the DBN corresponds to one agent. (Right) A graphical depiction of the influence effects in a multi-agent MDP. A connection between nodes in the graph implies arrows connecting the nodes in the DBN.

Our goal is to characterize the scaling of the sample complexity for various reinforcement learning approaches (i.e., how many trials they require in order to learn a near-optimal policy) for large numbers of agents $n$. Thus, in our bounds below, no serious attempt has been made to make our bounds tight in variables other than $n$.

## 3 Global rewards hardness result

Below we show that if an RL algorithm uses only the global reward signal, then there exists a very simple MDP—one with horizon, $T = 1$, only one state/trivial dynamics, and two actions per agent—on which the learning algorithm will require $\tilde{\Omega}(n)$ trials to learn a good policy. Thus, such algorithms do not scale well to large numbers of agents. For example, consider learning in the traffic signal problem described in the introduction with $n = 100,000$ traffic lights. Such an algorithm may then require on the order of $100,000$ days of experience (trials) to learn. In contrast, in Section 4, we show that if a reinforcement learning algorithm is given access to the local rewards, it can be possible to learn in such problems with an exponentially smaller $O(\log n)$ sample complexity.

**Theorem 3.1:** *Let any $0 < \epsilon < 0.05$ be fixed. Let any reinforcement learning algorithm $\mathcal{L}$ be given that only uses the* global reward *signal $R(s)$, and does not use the local rewards $R_i(s^{(i)})$ to learn (other than through their sum). Then there exists an MDP with time horizon $T = 1$, so that:*

1. *The MDP is very "simple" in that it has only one state ($|S| = 1$, $|S^n| = 1$); trivial state transition probabilities (since $T = 1$); two actions per agent ($|A| = 2$); and deterministic binary (0/1)-valued local reward functions.*

2. *In order for $\mathcal{L}$ to output a policy $\hat{\pi}$ that is near-optimal satisfying[2] $U(\hat{\pi}) \geq \max_\pi U(\pi) - \epsilon$,it is necessary that the number of trials $m$ be at least*

$$m \geq \frac{0.32n + \log(1/4)}{\log(n+1)} = \tilde{\Omega}(n).$$

**Proof.** For simplicity, we first assume that $\mathcal{L}$ is a deterministic learning algorithm, so that in each of the $m$ trials, its choice of action is some deterministic function of the outcomes of the earlier trials. Thus, in each of the $m$ trials, $\mathcal{L}$ chooses a vector of actions $a \in A^N$, and receives the global reward signal $R(s, a) = \frac{1}{n} \sum_{i=1}^n R(s^{(i)}, a^{(i)})$. In our MDP, each local reward $R(s^{(i)}, a^{(i)})$ will take values only 0 and 1. Thus, $R(s, a)$ can take only $n + 1$ different values (namely, $\frac{0}{n}, \frac{1}{n}, \dots, \frac{n}{n}$). Since $T = 1$, the algorithm receives only one such reward value in each trial.

Let $r_1, \dots, r_m$ be the $m$ global reward signals received by $\mathcal{L}$ in the $m$ trials. Since $\mathcal{L}$ is deterministic, its output policy $\hat{\pi}$ will be chosen as some deterministic function of these

rewards $r_1, \ldots, r_m$. But the vector $(r_1, \ldots, r_m)$ can take on only $(n+1)^m$ different values (since each $r_t$ can take only $n+1$ different values), and thus $\hat{\pi}$ itself can also take only at most $(n+1)^m$ different values. Let $\Pi_m$ denote this set of possible values for $\hat{\pi}$. ($|\Pi_m| \leq (n+1)^m$).

Call each local agent's two actions $a_1, a_2$. We will generate an MDP with randomly chosen parameters. Specifically, each local reward $R_i(s^{(i)}, a^{(i)})$ function is randomly chosen with equal probability to either give reward 1 for action $a_1$ and reward 0 for action $a_2$; or vice versa. Thus, each local agent has one "right" action that gives reward 1, but the algorithm has to learn which of the two actions this is. Further, by choosing the right actions, the optimal policy $\pi^*$ attains $U(\pi^*) = 1$.

Fix any policy $\pi$. Then $U_M(\pi) = \frac{1}{n} \sum_{i=1}^{n} R(s^{(i)}, \pi(s^{(i)}))$ is the mean of $n$ independent Bernoulli(0.5) random variables (since the rewards are chosen randomly), and has expected value 0.5. Thus, by the Hoeffding inequality, $P(U_M(\pi) \geq 1-2\epsilon) \leq \exp(-2(0.5-2\epsilon)^2 n)$. Thus, taking a union bound over all policies $\pi \in \Pi_M$, we have

$$P(\exists \pi \in \Pi_M \text{ s.t. } U_M(\pi) \geq 1 - 2\epsilon) \quad \leq \quad |\Pi_M| \exp(-2(0.5 - 2\epsilon)^2 n) \qquad (2)$$
$$\leq \quad (n+1)^m \exp(-2(0.5 - 2\epsilon)^2 n) \qquad (3)$$

Here, the probability is over the random MDP $M$. But since $\mathcal{L}$ outputs a policy in $\Pi_M$, the chance of $\mathcal{L}$ outputting a policy $\hat{\pi}$ with $U_M(\hat{\pi}) \geq 1 - 2\epsilon$ is bounded by the chance that there exists such a policy in $\Pi_M$. Thus,

$$P(U_M(\hat{\pi}) \geq 1 - 2\epsilon) \leq (n+1)^m \exp(-2(0.5 - 2\epsilon)^2 n). \qquad (4)$$

By setting the right hand side to $1/4$ and solving for $m$, we see that so long as

$$m < \frac{2(0.5 - 2\epsilon)^2 n + \log(1/4)}{\log(n+1)} \leq \frac{0.32n + \log(1/4)}{\log(n+1)}, \qquad (5)$$

we have that $P(U_M(\hat{\pi}) \geq 1 - 2\epsilon) < 1/4$. (The second equality above follows by taking $\epsilon < 0.05$, ensuring that no policy will be within 0.1 of optimal.) Thus, under this condition, by the standard probabilistic method argument [1], there must be at least one such MDP under which $\mathcal{L}$ fails to find an $\epsilon$-optimal policy.

For randomized algorithms $\mathcal{L}$, we can define for each string of input random numbers to the algorithm $\omega$ a deterministic algorithm $\mathcal{L}^\omega$. Given $m$ samples above, the expected performance of algorithm $\mathcal{L}^\omega$ over the distribution of MDPs

$$E_{p(M)}[\mathcal{L}^\omega] \quad \leq \quad Pr(U_M(\mathcal{L}^\omega) \geq 1 - 2\epsilon)1 + (1 - Pr(U_M(\mathcal{L}^\omega) \geq 1 - 2\epsilon))(1 - 2\epsilon)$$
$$< \quad \frac{1}{4} + \frac{3}{4}(1 - 2\epsilon) < 1 - \epsilon$$

Since

$$E_{p(M)}E_{p(\omega)}[U_M(\mathcal{L}^\omega)] = E_{p(\omega)}E_{p(M)}[U_M(\mathcal{L}^\omega)] < E_{p(\omega)}[1 - \epsilon]$$

it follows again from the probabilistic method there must be at least one MDP for which the $\mathcal{L}$ has expected performance less than $1 - \epsilon$. $\qquad \square$

## 4 Learning with local rewards

Assuming the existence of a good exploration policy, we now show a positive result that if our learning algorithm has access to the local rewards, then it is possible to learn a near-optimal policy after a number of trials that grows only *logarithmically* in the number of agents $n$. In this section, we will assume that the neighborhood structure (encoded by $\text{neigh}(i)$) is known, but that the CPT parameters of the dynamics and the reward functions are unknown. We also assume that the size of the largest neighborhood is bounded by $\max_i |\text{neigh}(i)| = B$.

**Definition.** A policy $\pi_{\text{explore}}$ is a $(\rho, \nu)$-exploration policy if, given any $i$, any configuration of states $s^{(\text{neigh}(i))} \in S^{|\text{neigh}(i)|}$, and any action $a^{(i)} \in A$, on a trial of length $T$ the policy $\pi_{\text{explore}}$ has at least a probability $\nu \cdot \rho^B$ of executing action $a^{(i)}$ while $i$ and its neighbors are in state $s^{(\text{neigh}(i))}$.

**Proposition 4.1:** *Suppose the MDP's initial state distribution is random, so that the state $s_i^{(i)}$ of each agent $i$ is chosen independently from some distribution $D_i$. Further, assume that $D_i$ assigns probability at least $\rho > 0$ to each possible state value $s \in S$. Then the "random" policy $\pi$ (that on each time-step chooses each agent's action uniformly at random over $A$) is a $(\rho, \frac{1}{|A|})$-exploration policy.*

**Proof.** For any agent $i$, the initial state of $s^{(\mathrm{neigh}(i))}$ has has at least a $\rho^B$ chance of being any particular vector of values, and the random action policy has a $1/|A|$ chance of taking any particular action from this state. $\qquad\square$

In general, it is a fairly strong assumption to assume that we have an exploration policy. However, this assumption serves to decouple the problem of exploration from the "sample complexity" question of how much data we need from the MDP. Specifically, it guarantees that we visit each local configuration sufficiently often to have a reasonable amount of data to estimate each CPT. [3]

In the envisioned procedure, we will execute an exploration policy for $m$ trials, and then use the resulting data we collect to obtain the maximum-likelihood estimates for the CPT entries and the rewards. We call the resulting estimates $\hat{p}(s_{t+1}^{(i)}|s_t^{(\mathrm{neigh}(i))}, a_t^{(i)})$ and $\hat{R}(s^{(i)}, a^{(i)})$.[4] The following simple lemma shows that, with a number of trials that grows only logarithmically in $n$, this procedure will give us good estimates for all CPTs and local rewards.

**Lemma 4.2:** *Let any $\epsilon_0 > 0, \delta > 0$ be fixed. Suppose $|\mathrm{neigh}(i)| \leq B$ for all $i$, and let a $(\rho, \nu)$-exploration policy be executed for $m$ trials. Then in order to guarantee that, with probability at least $1 - \delta$, the CPT and reward estimates are $\epsilon_0$-accurate:*

$$|\hat{p}(s_{t+1}^{(i)}|s_t^{(\mathrm{neigh}(i))}, a_t^{(i)}) - p(s_{t+1}^{(i)}|s_t^{(\mathrm{neigh}(i))}, a_t^{(i)})| \leq \epsilon_0 \qquad \text{for all } i, s_{t+1}^{(i)}, s_t^{(\mathrm{neigh}(i))}, a_t^{(i)}$$

$$|\hat{R}(s^{(i)}, a^{(i)})| - R(s^{(i)}, a^{(i)})| \leq \epsilon_0 \qquad \text{for all } i, s^{(i)}, a^{(i)}, \qquad (6)$$

*it suffices that the number of trials be*

$$m = O((\log n) \cdot \mathrm{poly}(\frac{1}{\epsilon_0}, \frac{1}{\delta}, |S|, |A|, 1/(\nu\rho^B), B, T)).$$

**Proof (Sketch).** Given $c$ examples to estimate a particular CPT entry (or a reward table entry), the probability that this estimate differs from the true value by more than $\epsilon_0$ can be controlled by the Hoeffding bound:

$$P(|\hat{p}(s_{t+1}^{(i)}|s_t^{(\mathrm{neigh}(i))}, a_t^{(i)}) - p(s_{t+1}^{(i)}|s_t^{(\mathrm{neigh}(i))}, a_t^{(i)})| \geq \epsilon_0) \leq 2\exp(-2\epsilon_0^2 c).$$

Each CPT has at most $|A||S|^{B+1}$ entries and there are $n$ such tables. There are also $n|S||A|$ possible local reward values. Taking a union bound over them, setting our probability of incorrectly estimating any CPTs or rewards to $\delta/2$, and solving for $c$ gives $c \geq \frac{2}{\epsilon_0^2}\log(\frac{4\,n\,|A||S|^{B+1}}{\delta})$. For each agent $i$ we see each local configurations of states and actions $(s^{(\mathrm{neigh}(i))}, a^{(i)})$ with probability $\geq \rho^B \nu$. For $m$ trajectories the expected number

of samples we see for each CPT entry is at least $m\rho^B\nu$. Call $S_m^{(s^{(\text{neigh}(i))}, a^{(i)})}$ the number of samples we've seen of a configuration $(s^{(\text{neigh}(i))}, a^{(i)})$ in $m$ trajectories. Note then that:

$$P(S_m^{(s^{(\text{neigh}(i))}, a^{(i)})} \leq c) \leq P(S_m^{(s^{(\text{neigh}(i))}, a^{(i)})} - E[S_m^{(s^{(\text{neigh}(i))}, a^{(i)})}] \leq c - m\rho^B\nu).$$

and another application of Hoeffding's bound ensures that:

$$P(S_m^{(s^{(\text{neigh}(i))}, a^{(i)})} - E[S_m^{(s^{(\text{neigh}(i))}, a^{(i)})}] \leq c - m\rho^B\nu) \leq \exp(\frac{-2}{mT^2}(c - m\rho^B\nu)^2).$$

Applying again the union bound to ensure that the probability of failure here is $\leq \delta/2$ and solving for $m$ gives the result. $\square$

**Definition.** Define the **radius of influence** $r(t)$ after $t$ steps to be the maximum number of nodes that are within $t$ steps in the neighborhood graph of any single node.

Viewed differently, $r(t)$ upper bounds the number of nodes in the $t$-th timeslice of the DBN (as in Figure 1) which are decendants of any single node in the 1-st timeslice. In a DBN as shown in Figure 1, we have $r(t) = O(t)$. If the neighborhood graph is a 2-d lattice in which each node has at most 4 neighbors, then $r(t) = O(t^2)$. More generally, we might expect to have $r(t) = O(t^2)$ for "most" planar neigborhood graphs. Note that, even in the worst case, by our assumption of each node having $B$ neighbors, we still have the bound $r(t) \leq B^t$, which is a bound independent of the number of agents $n$.

**Theorem 4.3:** *Let any $\epsilon > 0, \delta > 0$ be fixed. Suppose $|\text{neigh}(i)| \leq B$ for all $i$, and let a $(\rho, \nu)$-exploration policy be executed for $m$ trials in the MDP $M$. Let $\hat{M}$ be the maximum likelihood MDP, estimated from data from these $m$ trials. Let $\Pi$ be a policy class, and let*

$$\hat{\pi} = \arg\max_{\pi \in \Pi} U_{\hat{M}}(\pi)$$

*be the best policy in the class, as evaluated on $\hat{M}$. Then to ensure that, with probability $1 - \delta$, we have that $\hat{\pi}$ is near-optimal within $\Pi$, i.e., that*

$$U_M(\hat{\pi}) \geq \max_{\pi \in \Pi} U_M(\pi) - \epsilon,$$

*it suffices that the number of trials be:*

$$m = O((\log n) \cdot \text{poly}(1/\epsilon, 1/\delta, |S|, |A|, 1/(\nu\rho^B)), B, T, r(T)).$$

**Proof.** Our approach is essentially constructive: we show that for any policy, finite-horizon value-iteration using approximate CPTs and rewards in its backups will correctly estimate the true value function for that policy within $\epsilon/2$. For simplicity, we assume that the initial state distribution is known (and thus the same in $\hat{M}$ and $M$); the generalization offers no difficulties. By lemma (4.2) with $m$ samples we can know both CPTs and rewards with the probability required within any required $\epsilon_0$.

Note also that for any MDP with the given DBN or neighborhood graph structure (including both $M$ and $\hat{M}$) the value function for every policy $\pi$ and at each time-step has a property of *bounded variation*:

$$|\hat{V}_t(s^{(1)}, \ldots s^{(n)}) - \hat{V}_t(s^{(1)}, \ldots s^{(i-1)}, s^{(i)}_{\text{changed}}, s^{(i+1)}, \ldots, s^{(n)}| \leq \frac{r(T)T}{n}$$

This follows since a change in state can effect at most $r(T)$ agents' states, so the resulting change in utility must be bounded by $r(T)T/n$.

To compute a bound on the error in our estimate of overall utility we compute a bound on the error induced by a one-step Bellman backup $||B\hat{V} - \hat{B}\hat{V}||_\infty$. This quantity can be bounded in turn by considering the sequence of partially correct backup operators $\hat{B}_0, \ldots, \hat{B}_n$ where $\hat{B}_i$ is defined as the Bellman operator for policy $\pi$ using the exact transitions and rewards for agents $1, 2, \ldots, i$, and the estimated transitions rewards/transitions

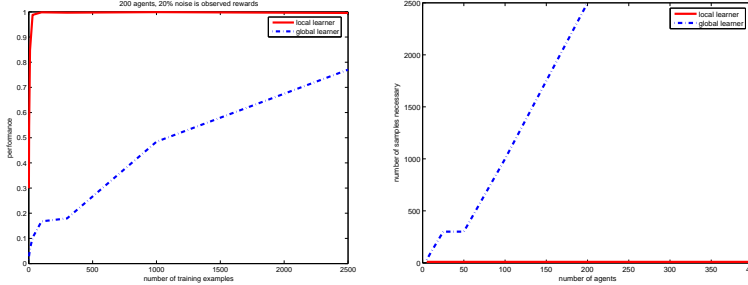

Figure 2: (Left) Scaling of performance as a function of the number of trajectories seen for a global reward and local reward algorithms. (Right) Scaling of the number of samples necessary to achieve near optimal reward as a function of the number of agents.

for agents $i+1, \ldots, n$. From this definition it is immediate that the total error is equivalent to the telescoping sum:

$$||B\hat{V} - \hat{B}\hat{V}||_\infty = ||\hat{B}_0\hat{V} - \hat{B}_1\hat{V} + \hat{B}_1\hat{V} - \ldots + \hat{B}_{n-1}\hat{V} - \hat{B}_n\hat{V}||_\infty \quad (7)$$

That sum is upper-bounded by the sum of term-by-term errors $\sum_{i=0}^{n-1} ||\hat{B}_i\hat{V} - \hat{B}_{i+1}\hat{V}||_\infty$. We can show that each of the terms in the sum is less than $\epsilon_0 r(T)(T+1)/n$ since the Bellman operators $\hat{B}_i\hat{V} - \hat{B}_{i+1}\hat{V}$ differ in the immediate reward contribution of agent $i+1$ by $\leq \epsilon_0$ and differ in computing the expected value of the future value by

$$E_{\prod_{j=1}^{i+1} p(s_{t+1}^j|s_t,\pi) \prod_{j=i+2}^{n} p(s_{t+1}^j|s_t,\pi)} \Big[ \sum_{s^{i+1}} \Delta p(s_{t+1}^{i+1}|s_t,\pi)\hat{V}_{t+1}(s) \Big],$$

with $\Delta p(s_{t+1}^{i+1}|s_t,\pi) \leq \epsilon_0$ the difference in the CPTs between $\hat{B}_i$ and $\hat{B}_{i+1}$. By the bounded variation argument this total is then less than $\epsilon_0 r(T)T|S|/n$. It follows then $\sum_i ||\hat{B}_i\hat{V} - \hat{B}_{i+1}\hat{V}||_\infty \leq \epsilon_0 \ r(T) \ (T+1)|S|$. We now appeal to finite-horizon bounds on the error induced by Bellman backups [11] to show that the $||\hat{V} - V||_\infty \leq T||B\hat{V} - \hat{B}\hat{V}||_\infty \leq T(T+1) \ \epsilon_0 \ r(T)|S|$. Taking the expectation of $\hat{V}$ with respect to the initial state distribution $D$ and setting $m$ according to Lemma (4.2) with $\epsilon_0 = \frac{\epsilon}{2|S|r(T) \ T(T+1)}$ completes the proof. $\qquad \square$

## 5 Demonstration

We first present an experimental domain that hews closely to the theory in Section (3) above to demonstrate the importance of local rewards. In our simple problem there are $n = 400$ independent agents who each choose an action in $\{0, 1\}$. Each agent has a "correct" action that earns it reward $R_i = 1$ with probability $0.8$, and reward $0$ with probability $0.2$. Equally, if the agents chooses the wrong action, it earns reward $R_i = 1$ with probability $0.2$.

We compare two methods on this problem. Our first *global* algorithm uses only the global rewards $R$ and uses this to build a model of the local rewards, and finally solves the resulting estimated MDP exactly. The local reward functions are learnt by a least-squares procedure with basis functions for each agent. The second algorithm also learns a local reward function, but does so taking advantage of the local rewards it observes as opposed to only the global signal. Figure (2) demonstrates the advantages of learning using a global reward signal.[5] On the right in Figure (2), we compute the time required to achieve $\frac{1}{4}$ of optimal reward for each algorithm, as a function of the number of agents.

In our next example, we consider a simple variant of the multi-agent SYSADMIN[6] prob-

lem [4]. Again, we consider two algorithms: a global REINFORCE [9] learner, and a RE-
INFORCE algorithm run using only local rewards, even through the local REINFORCE al-
gorithm run in this way is not guaranteed to converge to the globally optimal (cooperative)
solution. We note that the local algorithm learns much more quickly than using the global
reward. (Figure 3) The learning speed we observed for the global algorithm correlates well
with the observations in [5] that the number of samples needed scales roughly linearly in
the number of agents. The local algorithm continued to require essentially the same number
of examples for all sizes used (up to over 100 agents) in our experiments.

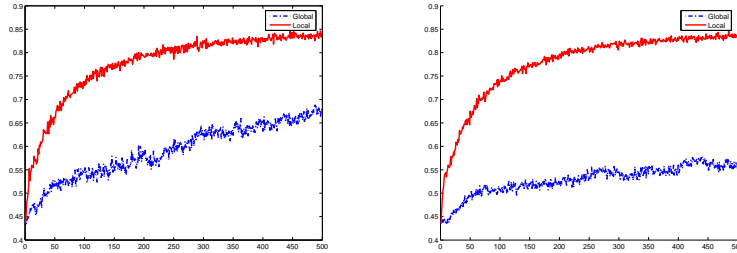

Figure 3: REINFORCE applied to the multi-agent SYSADMIN problem. *Local* refers to REINFORCE
applied using only neighborhood (local) rewards while *global* refers to standard REINFORCE (applied
to the global reward signal). (Left) shows averaged reward performance as a function of number of
iterations for 10 agents. (Right) depicts the performance for 20 agents.

## Footnotes

[1] Big-$\tilde{\Omega}$ notation omits logarithmic terms, similar to how big-$\Omega$ notation drops constant values.

[2]For randomized algorithms we consider instead the expectation of $U(\hat{\pi})$ under the algorithm's randomization.

[3]Further, it is possible to show a stronger version of our result than that stated below, showing that a random action policy can always be used as our exploration policy, to obtain a sample complexity bound with the same logarithmic dependence on $n$ (but significantly worse dependencies on $T$ and $B$). This result uses ideas from the random trajectory method of [8], with the key observation that local configurations that are not visited reasonably frequently by the random exploration policy will not be visited frequently by *any* policy, and thus inaccuracies in our estimates of their CPT entries will not significantly affect the result.

[4]We let $\hat{p}(s_{t+1}^{(i)}|s_t^{(\mathrm{neigh}(i))}, a_t^{(i)})$ be the uniform distribution if $(s_t^{(\mathrm{neigh}(i))}, a_t^{(i)})$ was never observed in the training data, and similarly let $\hat{R}(s^{(i)}, a^{(i)}) = 0$ if $\hat{R}(s^{(i)}, a^{(i)})$ was never observed.

[5]A gradient-based model-free approach using the global reward signal was also tried, but its performance was significantly poorer than that of the two algorithms depicted in Figure (2, left).

[6]In SYSADMIN there is a network of computers that fail randomly. A computer is more likely to fail if a neighboring computer (arranged in a ring topology) fails. The goal is to reboot machines in such a fashion so a maximize the number of running computers.

# References

[1]   N. Alon and J. Spencer. *The Probabilistic Method*. Wiley, 2000.

[2]   C. Boutilier, T. Dean, and S. Hanks. Decision theoretic planning: Structural assumptions and
      computational leverage. *Journal of Artificial Intelligence Research*, 1999.

[3]   Y. Chang, T. Ho, and L. Kaelbling. All learning is local: Multi-agent learning in global reward
      games. In *Advances in NIPS 14*, 2004.

[4]   C. Guestrin, D. Koller, and R. Parr. Multi-agent planning with factored MDPs. In *NIPS-14*,
      2002.

[5]   C. Guestrin, M. Lagoudakis, and R. Parr. Coordinated reinforcement learning. In *ICML*, 2002.

[6]   M. Kearns and D. Koller. Efficient reinforcement learning in factored mdps. In *IJCAI 16*, 1999.

[7]   M. Kearns, M. Littman, and S. Singh. Graphical models for game theory. In *UAI*, 2001.

[8]   M. Kearns, Y. Mansour, and A. Ng. Approximate planning in large POMDPs via reusable
      trajectories. *(extended version of paper in NIPS 12)*, 1999.

[9]   L. Peshkin, K-E. Kim, N. Meleau, and L. Kaelbling. Learning to cooperate via policy search.
      In *UAI 16*, 2000.

[10]  J. Schneider, W. Wong, A. Moore, and M. Riedmiller. Distributed value functions. In *ICML*,
      1999.

[11]  R. Williams and L. Baird. Tight performance bounds on greedy policies based on imperfect
      value functions. Technical report, Northeastern University, 1993.
